# Learning To Count Objects in Images

**Victor Lempitsky**
Visual Geometry Group
University of Oxford

**Andrew Zisserman**
Visual Geometry Group
University of Oxford

## Abstract

We propose a new supervised learning framework for visual object counting tasks, such as estimating the number of cells in a microscopic image or the number of humans in surveillance video frames. We focus on the practically-attractive case when the training images are annotated with dots (one dot per object).

Our goal is to accurately estimate the count. However, we evade the hard task of learning to detect and localize individual object instances. Instead, we cast the problem as that of estimating an image density whose integral over any image region gives the count of objects within that region. Learning to infer such density can be formulated as a minimization of a regularized risk quadratic cost function. We introduce a new loss function, which is well-suited for such learning, and at the same time can be computed efficiently via a maximum subarray algorithm. The learning can then be posed as a convex quadratic program solvable with cutting-plane optimization.

The proposed framework is very flexible as it can accept any domain-specific visual features. Once trained, our system provides accurate object counts and requires a very small time overhead over the feature extraction step, making it a good candidate for applications involving real-time processing or dealing with huge amount of visual data.

## 1 Introduction

The *counting* problem is the estimation of the number of objects in a still image or video frame. It arises in many real-world applications including cell counting in microscopic images, monitoring crowds in surveillance systems, and performing wildlife census or counting the number of trees in an aerial image of a forest.

We take a supervised learning approach to this problem, and so require a set of training images with annotation. The question is what level of annotation is required? Arguably, the bare minimum of annotation is to provide the overall count of objects in each training image. This paper focusses on the next level of annotation which is to specify the object position by putting a single dot on each object instance in each image. Figure 1 gives examples of the counting problems and the dotted annotation we consider.

Dotting (pointing) is the natural way to count objects for humans, at least when the number of objects is large. It may be argued therefore that providing dotted annotations for the training images is no harder for a human than giving just the raw counts. On the other hand, a spatial arrangement of the dots provides a wealth of additional information, and this paper is, in part, about how to exploit this "free lunch" (in the context of the counting problem). Overall, it should be noted that dotted annotation is less labour-intensive than the bounding-box annotation, let alone pixel-accurate annotation, traditionally used by the supervised methods in the computer vision community [15]. Therefore, the dotted annotation represents an interesting and, perhaps, under-investigated case.

This paper develops a simple and general discriminative learning-based framework for counting objects in images. Similar to global regression methods (see below), it also evades the hard problem of detecting all object instances in the images. However, unlike such methods, the approach also takes full and extensive use of the spatial information contained in the dotted supervision.

The high-level idea of our approach is extremely simple: given an image $I$, our goal is to recover a *density function $F$* as a real function of pixels in this image. Our notion of density function loosely

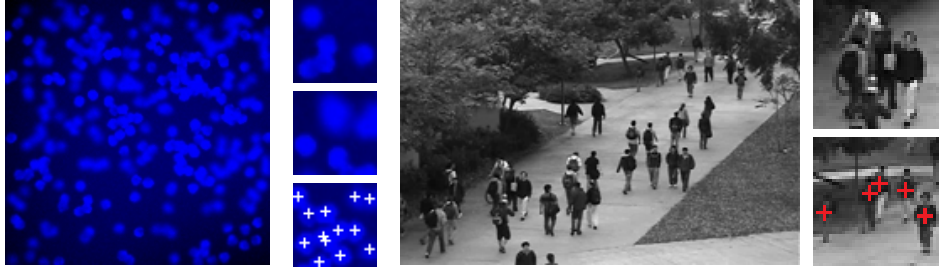

Figure 1: **Examples of counting problems**. *Left* — counting bacterial cells in a fluorescence-light microscopy image (from [29]), *right* — counting people in a surveillance video frame (from [10]). Close-ups are shown alongside the images. The bottom close-ups show examples of the dotted annotations (crosses). Our framework learns to estimate the number of objects in the previously unseen images based on a set of training images of the same kind augmented with dotted annotations.

corresponds to the physical notion of density as well as to the mathematical notion of measure. Given the estimate $F$ of the density function and the query about the number of objects in the entire image $I$, the number of objects in the image is estimated by integrating $F$ over the entire $I$. Furthermore, integrating the density over an image subregion $S \subset I$ gives an estimate of the count of objects in that subregion.

Our approach assumes that each pixel $p$ in an image is represented by a feature vector $x_p$ and models the density function as a linear transformation of $x_p$: $F(p) = w^T x_p$. Given a set of training images, the parameter vector $w$ is learnt in the regularized risk framework, so that the density function estimates for the training images matches the ground truth densities inferred from the user annotations (under regularization on $w$).

The key conceptual difficulty with the density function is the discrete nature of both image observations (pixel grid) and, in particular, the user training annotation (sparse set of dots). As a result, while it is easy to reason about average densities over the extended image regions (e.g. the whole image), the notion of density is not well-defined at a pixel level. Thus, given a set of dotted annotation there is no trivial answer to the question: what should be the ground truth density for this training example. Consequently, this local ambiguity also renders standard pixel-based distances between density functions inappropriate for the regularized risk framework.

Our main contribution, addressing this conceptual difficulty, is a specific distance metric $\mathcal{D}$ between density functions used as a loss in our framework, which we call the *MESA* distance (where MESA stands for *Maximum Excess over SubArrays*, as well as for the geological term for the elevated plateau). This distance possess two highly desirable properties:

*1. Robustness.* The MESA distance is robust to the additive local perturbations of its arguments such as independent noise or high-frequency signal as long as the integrals (counts) of these perturbations over larger region are close to zero. Thus, it does not matter much how exactly we define the ground truth density locally, as long as the integrals of the ground truth density over the larger regions reflect the counts correctly. We can then naturally define the "ground truth" density for a dotted annotation to be a sum of normalized gaussians centered at the dots.

*2. Computability.* The MESA distance can be computed exactly via an efficient combinatorial algorithm (maximum sub-array [8]). Plugging it into the regularized risk framework then leads to a convex quadratic program for estimating $w$. While this program has a combinatorial number of linear constraints, the cutting-plane procedure finds the close approximation to the globally optimal $w$ after a small number of iterations.

The proposed approach is highly versatile. As virtually no assumptions is made about the features $x_p$, our framework can benefit from much of the research on good features for object detection. Thus, the confidence maps produced by object detectors or the scene explanations resulting from fitting the generative models can be turned into features and used by our method.

## 1.1 Related work.

A number of approaches tackle counting problems in an unsupervised way, performing grouping based on self-similarities [3] or motion similarities [27]. However, the counting accuracy of such fully unsupervised methods is limited, and therefore others considered approaches based on supervised learning. Those fall into two categories:

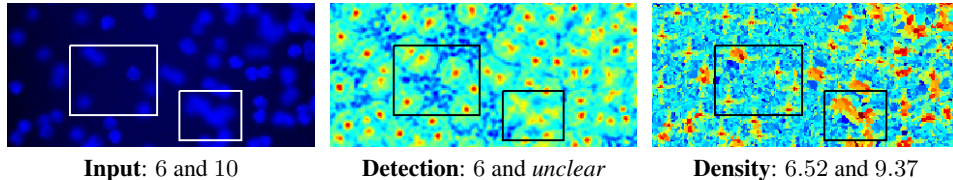

**Input**: 6 and 10      **Detection**: 6 and *unclear*      **Density**: 6.52 and 9.37

Figure 2: **Processing results for a previously unseen image**. *Left* – a fragment of the microscopy image. Emphasized are the two rectangles containing 6 and 10 cells respectively. *Middle* – the confidence map produced by an SVM-based detector, 6 peaks are clearly discernible for the 1st rectangle, but the number of peaks in the 2nd rectangle is unclear. *Right* – the density map, that our approach produces. The integrals over the rectangles (6.52 and 9.37) are close to the correct number of cells. *(MATLAB jet colormap is used)*

**Counting by detection:** This assumes the use of a visual object detector, that localizes individual object instances in the image. Given the localizations of all instances, counting becomes trivial. However, object detection is very far from being solved [15], especially for overlapping instances. In particular, most current object detectors operate in two stages: first producing a real-valued confidence map; and second, given such a map, a further *thresholding* and *non-maximum suppression* steps are needed to locate peaks correspoinding to individual instances [12, 26]. More generative approaches avoid non-maximum suppression by reasoning about relations between object parts and instances [6, 14, 20, 33, 34], but they are still geared towards a situation with a small number of objects in images and require time-consuming inference. Alternatively, several methods assume that objects tend to be uniform and disconnected from each other by the distinct background color, so that it is possible to localize individual instances via a Monte-Carlo process [13], morphological analysis [5, 29] or variational optimization [25]. Methods in these groups deliver accurate counts when their underlying assumptions are met but are not applicable in more challenging situations.

**Counting by regression:** These methods avoid solving the hard detection problem. Instead, a direct mapping from some global image characteristics (mainly histograms of various features) to the number of objects is learned. Such a standard regression problem can be addressed by a multitude of machine learning tools (e.g. neural networks [11, 17, 22]). This approach however has to discard any available information about the location of the objects (dots), using only its 1-dimensional statistics (total number) for learning. As a result, a large number of training images with the supplied counts needs to be provided during training. Finally, **counting by segmentation** methods [10, 28] can be regarded as hybrids of counting-by-detection and counting-by-regression approaches. They segment the objects into separate clusters and then regress from the global properties of each cluster to the overall number of objects in it.

## 2 The Framework

We now provide the detailed description of our framework starting with the description of the learning setting and notation.

### 2.1 Learning to Count

We assume that a set of $N$ training images (pixel grids) $I_1, I_2, \ldots I_N$ is given. It is also assumed that each pixel $p$ in each image $I_i$ is associated with a real-valued feature vector $x_p^i \in \mathbf{R}^K$. We give the examples of the particular choices of the feature vectors in the experimental section. It is finally assumed that each training image $I_i$ is annotated with a set of 2D points $\mathbf{P}_i = \{P_1, \ldots, P_{C(i)}\}$, where $C(i)$ is the total number of objects annotated by the user.

The density functions in our approaches are real-valued functions over pixel grids, whose integrals over image regions should match the object counts. For a training image $I_i$, we define the *ground truth* density function to be a kernel density estimate based on the provided points:

$$\forall p \in I_i, \quad F_i^0(p) = \sum_{P \in \mathbf{P}_i} \mathcal{N}(p; P, \sigma^2 \mathbf{1}_{2 \times 2}). \tag{1}$$

Here, $p$ denotes a pixel, $\mathcal{N}(p; P, \sigma^2 \mathbf{1}_{2 \times 2})$ denotes a normalized 2D Gaussian kernel evaluated at $p$, with the mean at the user-placed dot $P$, and an isotropic covariance matrix with $\sigma$ being a small value (typically, a few pixels). With this definition, the sum of the ground truth density $\sum_{p \in I_i} F_i^0(p)$ over the entire image will not match the dot count $C_i$ exactly, as dots that lie very close to the image boundary result in their Gaussian probability mass being partly outside the image. This is a natural and desirable

behaviour for most applications, as in many cases an object that lies partly outside the image boundary should not be counted as a full object, but rather as a fraction of an object.

Given a set of training images together with their ground truth densities, we aim to learn the linear transformation of the feature representation that approximates the density function at each pixel:

$$\forall p \in I_i, \quad F_i(p|w) = w^T x_p^i, \tag{2}$$

where $w \in R^K$ is the parameter vector of the linear transform that we aim to learn from the training data, and $F_i(\cdot|w)$ is the estimate of the density function for a particular value of $w$. The regularized risk framework then suggests choosing $w$ so that it minimizes the sum of the mismatches between the ground truth and the estimated density functions (the loss function) under regularization:

$$w = \operatorname*{argmin}_{w} \left( w^T w + \lambda \sum_{i=1}^{N} \mathcal{D}\left( F_i^0(\cdot), F_i(\cdot|w) \right) \right), \tag{3}$$

Here, $\lambda$ is a standard scalar hyperparameter, controlling the regularization strength. It is the only hyperparameter in our framework (in addition to those that might be used during feature extraction).

After the optimal weight vector has been learned from the training data, the system can produce a density estimate for an unseen image $I$ by a simple linear weighting of the feature vector computed in each pixel as suggested by (2). The problem is thus reduced to choosing the right loss function $\mathcal{D}$ and computing the optimal $w$ in (3) under that loss.

## 2.2   The MESA distance

The distance $\mathcal{D}$ in (3) measures the mismatch between the ground truth and the estimated densities (the loss) and has a significant impact on the performance of the entire learning framework. There are two natural choices for $\mathcal{D}$:

- One can choose $\mathcal{D}$ to be some function of an $L_P$ metric, e.g. the $L_1$ metric (sum of absolute per-pixel differences) or a square of the $L_2$ metric (sum of squared per-pixel differences). Such choices turns (3) into standard regression problems (i.e. support vector regression and ridge regression for $L_1$ and $L_2^2$ cases respectively), where each pixel in each training image effectively provides a sample in the training set. The problem with such loss is that it is not directly related to the real quantity that we care about, i.e. the overall counts of objects in images. E.g. strong zero-mean noise would affect such metric a lot, while the overall counts would be unaffected.

- As the overall counts is what we ultimately care about, one may choose $\mathcal{D}$ to be an absolute or squared difference between the overall sums over the entire images for the two arguments, e.g. $\mathcal{D}\left( F_1(\cdot), F_2(\cdot) \right) = |\sum_{p \in I} F_1(p) - \sum_{p \in I} F_2(p)|$. The use of such a pseudometric as a loss turns (3) into the counting-by-regression framework discussed in Section 1.1. Once again, we get either the support vector regression (for the absolute differences) or ridge regression (for the squared differences), but now each training sample corresponds to the entire training image. Thus, although this choice of the loss matches our ultimate goal of learning to count very well, it requires many annotated images for training as spatial information in the annotation is discarded.

Given the significant drawbacks of both baseline distance measures, we suggest an alternative, which we call the MESA distance. Given an image $I$, the MESA distance $\mathcal{D}_{MESA}$ between two functions $F_1(p)$ and $F_2(p)$ on the pixel grid is defined as the largest absolute difference between sums of $F_1(p)$ and $F_2(p)$ over all box subarrays in $I$:

$$\mathcal{D}_{MESA}(F_1, F_2) = \max_{B \in \mathbf{B}} \left| \sum_{p \in B} F_1(p) - \sum_{p \in B} F_2(p) \right| \tag{4}$$

Here, $\mathbf{B}$ is the set of all box subarrays of $I$.

The MESA distance (in fact, a metric) can be regarded as an $L_\infty$ distance between combinatorially-long vectors of subarray sums. In the 1D case, it is related to the Kolmogorov-Smirnov distance between probability distributions [23] (in our terminology, the Kolmogorov-Smirnov distance is the maximum of absolute differences over the subarrays with one corner fixed at top-left; thus the strict subset of $\mathbf{B}$ is considered in the Kolmogorov-Smirnov case).

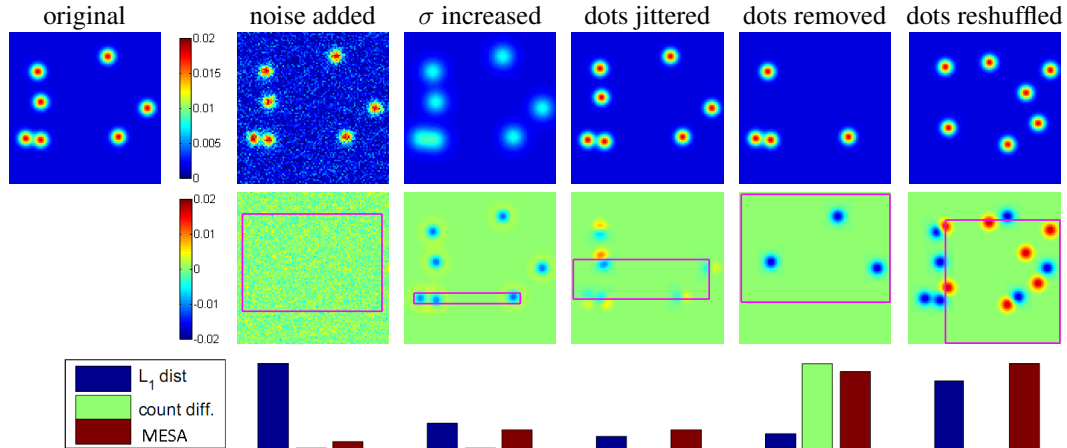

Figure 3: **Comparison of distances for matching density functions**. Here, the *top-left* image shows one of the densities, computed as the ground truth density for a set of dots. The densities in the *top row* are obtained through some perturbations of the original one. In the *bottom row*, we compare side-by-side the per-pixel $L_1$ distance, the absolute difference of overall counts, and the MESA distance between the original and the perturbed densities (the distances are normalized across the 5 examples). The MESA distance has a unique property that it tolerates the local modifications (noise, jitter, change of Gaussian kernel), but reacts strongly to the change in the number of objects or their positions. In the *middle row* we give per-pixel plots of the differences between the respective densities and show the boxes on which the maxima in the definition of the MESA distance are achieved.

The MESA distance has a number of desirable properties in our framework. Firstly, it is directly related to the counting objective we want to optimize. Since the set of all subarrays include the full image, $D_{MESA}(F_1, F_2)$ is an upper bound on the absolute difference of the overall count estimates given by the two densities $F_1$ and $F_2$. Secondly, when the two density functions differ by a zero-mean high-frequency signal or an independent zero-mean noise, the $D_{MESA}$ distance between them is small, because positive and negative deviations of $F_1$ from $F_2$ pixels tend to cancel each other over the large regions. Thirdly, $D_{MESA}$ is sensitive to the overall spatial layout of the denisities. Thus, if the difference between $F_1$ and $F_2$ is a low-frequency signal, e.g. $F_1$ and $F_2$ are the ground truth densities corresponding to the two point sets leaning towards two different corners of the image, then the $D_{MESA}$ distance between $F_1$ and $F_2$ is large, even if $F_1$ and $F_2$ sum to the same counts over the entire image. These properties are illustrated in Figure 3.

The final property of $\mathcal{D}_{MESA}$ is that it can be computed efficiently. This is because it can be rewritten as:

$$\mathcal{D}_{MESA}(F_1, F_2) = \max \left( \max_{B \in \mathbf{B}} \sum_{p \in B} \Big( F_1(p) - F_2(p) \Big), \quad \max_{B \in \mathbf{B}} \sum_{p \in B} \Big( F_2(p) - F_1(p) \Big) \right). \quad (5)$$

Computing both inner maxima in (5) then constitutes a *2D maximum subarray problem*, which is finding the box subarray of a given 2D array with the largest sum. This problem has a number of efficient solutions. Perhaps, the simplest of the efficient ones (from [8]) is an exhaustive search over one image dimension (e.g. for the top and bottom dimensions of the optimal subarray) combined with the dynamic programming (*Kadane*'s algorithm [7]) to solve the 1D maximum subarray problem along the other dimension in the inner loop. This approach has complexity $O(|I|^{1.5})$, where $|I|$ is the number of pixels in the image grid. It can be further improved in practice by replacing the exhaustive search over the first dimension with branch-and-bound [4]. More extensive algorithms that guarantee even better worst-case complexity are known [31]. In our experiments, the algorithm [8] was sufficient, as the time bottleneck lied in the QP solver (see below).

## 2.3 Optimization

We finally discuss how the optimization problem in (3) can be solved in the case when the $\mathcal{D}_{MESA}$ distance is employed. The learning problem (3) can then be rewritten as a convex quadratic program:

$$\min_{w,\xi_1,\ldots\xi_N} \quad w^T w + \lambda \sum_{i=1}^{N} \xi_i, \qquad \text{subject to} \tag{6}$$

$$\forall i, \ \forall B \in \mathbf{B}_i: \quad \xi_i \geq \sum_{p \in B} \left( F_i^0(p) - w^T x_p^i \right), \quad \xi_i \geq \sum_{p \in B} \left( w^T x_p^i - F_i^0(p) \right) \tag{7}$$

Here, $\xi_i$ are the auxiliary slack variables (one for each training image) and $\mathbf{B}_i$ is the set of all subarrays in image $i$. At the optimum of (6)–(7), the optimal vector $\hat{w}$ is the solution of (3) while the slack variables equal the MESA distances: $\hat{\xi}_i = \mathcal{D}_{MESA}\left(F_i^0(\cdot), F_i(\cdot|\hat{w})\right)$.

The number of linear constraints in (7) is combinatorial, so that a custom QP-solver cannot be applied directly. A standard iterative cutting-plane procedure, however, overcomes this problem: one starts with only a small subset of constraints activated (we choose 20 boxes with random dimensions in random subset of images to initialize the process). At each iteration, the QP (6)–(7) is solved with an active subset of constraints. Given the solution $^j w, ^j \xi_1, \ldots ^j \xi_N$ after the iteration $j$, one can find the box subarrays corresponding to the most violated constraints among (7). To do that, for each image we find the subarrays that maximize the right hand sides of (7), which are exactly the 2D maximum subarrays of $F_i^0(\cdot) - F_i(\cdot|^j w)$ and $F_i(\cdot|^j w) - F_i^0(\cdot)$ respectively.

The boxes $^j B_i^1$ and $^j B_i^2$ corresponding to these maximum subarrays are then found for each image $i$. If the respective sums $\sum_{p \in ^j B_i^1} \left( F_i^0(p) - ^j w^T x_p^i \right)$ and $\sum_{p \in ^j B_i^2} \left( ^j w^T x_p^i - F_i^0(p) \right)$ exceed $^j \xi_i \cdot (1 + \epsilon)$, the corresponding constraints are activated, and the next iteration is performed. The iterations terminate when for all images the sums corresponding to maximum subarrays are within $(1 + \epsilon)$ factor from $^j \xi_i$ and hence no constraints are activated. In the derivation here, $\epsilon << 1$ is a constant that promotes convergence in a small number of iterations to the approximation of the global minimum. Setting $\epsilon$ to 0 solves the program (6)–(7) exactly, while it has been shown in similar circumstances [16] that setting $\epsilon$ to a small finite value does not affect the generalization of the learning algorithm and brings the guarantees of convergence in small number of steps.

## 3    Experiments

Our framework and several baselines were evaluated on counting tasks for two types of imagery shown in Figure 1. We now discuss the experiments and the quantitative results. The test datasets and the densities computed with our method can be further assessed qualitatively at the project webpage [1].

**Bacterial cells in fluorescence-light microscopy images.** Our first experiment is concerned with synthetic images, emulating microscopic views of the colonies of bacterial cell, generated with [19] (Figure 1-left). Such synthetic images (Figure 1-left) are highly realistic and simulate such effects as cell overlaps, shape variability, strong out-of-focus blur, vignetting, etc. For the experiments, we generated a dataset of images (available at [1]), with the overall number of cells varying between 74 and 317. Few annotated datasets with real cell microscopy images also exist. While it is tempting to use real rather than synthetic imagery, all the real image datasets to the best of our knowledge are small (only few images have annotations), and, most importantly, there always are very big discrepancies between the annotations of different human experts. The latter effectively invalidates the use of such real datasets for quantitative comparison of different counting approaches.

Below we discuss the comparison of the counting accuracy achieved by our approach and baseline approaches. The features used in all approaches were based on the dense SIFT descriptor [21] computing using [32] software at each pixel of each image with the fixed SIFT frame radius (about the size of the cell) and fixed orientation. Each algorithm was trained on $N$ training images, while another $N$ images were used for the validation of metaparameters. The following approaches were considered:

*1. The proposed density-based approach.* A very simple feature representation was chosen: a codebook of $K$ entries was constructed via k-means on SIFT descriptors extracted from the hold-out 20 images. Then each pixel is represented by a vector of length $K$, which is 1 at the dimension corresponding to the entry of the SIFT descriptor at that pixel and 0 for all other dimensions. We used training images to learn the vector $w$ as discussed in Section 2.1. Counting is then performed by summing the values $w_t$ assigned to the codebook entries $t$ for all pixels in the test image. Figure 2-right gives an example of the respective density (see also [1]).

| | Validation | $N=1$ | $N=2$ | $N=4$ | $N=8$ | $N=16$ | $N=32$ |
|---|---|---|---|---|---|---|---|
| linear ridge regression | counting | $67.3_{\pm25.2}$ | $37.7_{\pm14.0}$ | $16.7_{\pm3.1}$ | $8.8_{\pm1.5}$ | $6.4_{\pm0.7}$ | $5.9_{\pm0.5}$ |
| kernel ridge regression | counting | $60.4_{\pm16.5}$ | $38.7_{\pm17.0}$ | $18.6_{\pm5.0}$ | $10.4_{\pm2.5}$ | $6.0_{\pm0.8}$ | $5.2_{\pm0.3}$ |
| detection | counting | $28.0_{\pm20.6}$ | $20.8_{\pm5.8}$ | $13.6_{\pm1.5}$ | $10.2_{\pm1.9}$ | $10.4_{\pm1.2}$ | $8.5_{\pm0.5}$ |
| detection | detection | $20.8_{\pm3.8}$ | $20.1_{\pm5.5}$ | $15.7_{\pm2.0}$ | $15.0_{\pm4.1}$ | $11.8_{\pm3.1}$ | $12.0_{\pm0.8}$ |
| detection+correction | counting | – | $22.6_{\pm5.3}$ | $16.8_{\pm6.5}$ | $6.8_{\pm1.2}$ | $6.1_{\pm1.6}$ | $4.9_{\pm0.5}$ |
| **density learning** | counting | $\mathbf{12.7_{\pm7.3}}$ | $\mathbf{7.8_{\pm3.7}}$ | $\mathbf{5.0_{\pm0.5}}$ | $\mathbf{4.6_{\pm0.6}}$ | $\mathbf{4.2_{\pm0.4}}$ | $\mathbf{3.6_{\pm0.2}}$ |
| **density learning** | MESA | $\mathbf{9.5_{\pm6.1}}$ | $\mathbf{6.3_{\pm1.2}}$ | $\mathbf{4.9_{\pm0.6}}$ | $\mathbf{4.9_{\pm0.7}}$ | $\mathbf{3.8_{\pm0.2}}$ | $\mathbf{3.5_{\pm0.2}}$ |

Table 1: **Mean absolute errors for cell counting on the test set of 100 fluorescent microscopy images.** The rows correspond to the methods described in the text. The second column corresponds to the error measure used for learning meta-parameters on the validation set. The last 6 columns correspond to the numbers of images in the training and validation sets. The average number of cells is $171_{\pm64}$ per image. Standard deviations in the table correspond to 5 different draws of training and validation image sets. The proposed method (density learning) outperforms considerably the baseline approaches (including the application-specific baseline with the error rate = 16.2) for all sizes of the training set.

| | 'maximal' | 'downscale' | 'upscale' | 'minimal' | 'dense' | 'sparse' |
|---|---|---|---|---|---|---|
| Counting-by-Regression [17] | 2.07 | 2.66 | 2.78 | N/A | N/A | N/A |
| Counting-by-Regression [28] | 1.80 | 2.34 | 2.52 | 4.46 | N/A | N/A |
| Counting-by-Segmentation [28] | **1.53** | 1.64 | 1.84 | **1.31** | N/A | N/A |
| **Density learning** | 1.70 | **1.28** | **1.59** | 2.02 | $1.78_{\pm0.39}$ | $2.06_{\pm0.59}$ |

Table 2: **Mean absolute errors for people counting in the surveillance video [10].** The columns correspond to the four scenarios (splits) reproduced from [28] ('maximal','downscale','upscale','minimal') and for the two new sets of splits ('dense' and 'sparse'). Our method outperforms counting-by-regression methods and is competitive with the hybrid method in [28], which uses more detailed annotation.

*2. The counting-by-regression baseline.* Each of the training images was described by a global histogram of the entries occurrences for the same codebook as above. We then learned two types of regression (ridge regression with linear and Gaussian kernels) to the number of cells in the image.

*3. The counting-by-detection baseline.* We trained a detector based on a linear SVM classifier. The SIFT descriptors corresponding to the dotted pixels were considered positive examples. To sample negative examples, we built a Delaunay triangulation on the dots and took SIFT descriptors corresponding to the pixels at the middle of Delaunay edges. At detection time, we applied the SVM at each pixel, and then found peaks in the resulting confidence map (e.g. Figure 2-middle) via non-maximum suppression with the threshold $\tau$ and radius $\rho$ using the code [18]. We also considered a variant with the linear correction of the obtained number to account for systematic biases (*detection+correction*). The slope and the intercept of the correction for each combination of $\tau$, $\rho$, and regularization strength were estimated via robust regression on the union of the training and validation sets.

*4. Application-specific method [29].* We also evaluated the software specifically designed for analyzing cells in fluorescence-light images [29]. The counting algorithm here is based on adaptive thresholding and morphological analysis. For this baseline, we tuned the free parameter (cell division threshold) on the test set, and computed the mean absolute error, which was **16.2**.

The meta-parameters ($K$, regularization strengths, Gaussian kernel width for ridge regression, $\tau$ and $\rho$ for non-maximum suppression) were learned in each case on the validation set. The objective minimized during the validation was counting accuracy. For counting-by-detection, we also considered optimizing detection accuracy (computed via Hungarian matching with the ground truth), and, for our approach, we also considered minimizing the MESA distance with the ground truth density on the validation set.

The results for a different number $N$ of training and validation images are given in Table 1, based on 5 random draws of training and validation sets. A hold out set of 100 images was used for testing. The proposed method outperforms the baseline approaches for all sizes of the training set.

**Pedestrians in surveillance video.** Here we focus on a 2000-frames video dataset [10] from a camera overviewing a busy pedestrian street (Figure 1-right). The authors of [10] also provided the dotted ground truth for these frames, the position of the ground plane, and the region of interest, where the counts should be performed. Recently, [28] performed extensive experiments on the dataset and reported the performance of three approaches (two counting-by-regression including [17] and the hybrid approach: split into blobs, and regress the number for each blob). The hybrid approach in [28] required more

detailed annotations than dotting (see [28] for details). For the sake of comparison, we adhered to the experimental protocols described in [28], so that the performance of our method is directly comparable.

In particular, 4 train/test splits were suggested in [28]: 1) 'maximal': train on frames 600:5:1400 (in Matlab notation) 2) 'downscale': train on frames 1205:5:1600 (the most crowded) 3) 'upscale': train on frames 805:5:1100 (the least crowded) 4) 'minimal': train on frames 640:80:1360 (10 frames). Testing is performed on the frames outside the training range. For future reference, we also included two additional scenarios ('dense' and 'sparse') with multiple similar splits in each (permitting variance estimation). Both scenarios are based on splitting the 2000 frames into 5 contiguous chunks of 400 frames. In each of the two scenarios, we then performed training on one chunk and testing on the other 4. In the "dense" scenario we trained on 80 frames sampled from the training split with uniform spacing, while in the 'sparse' scenario, we took just 10 frames.

Extracting features in this case is more involved as several modalities, namely the image itself, the difference image with the previous frame, and the background subtracted image have to be combined to achieve the best performance (a simple median filtering was used to estimate the static background image). We used a randomized tree approach similar to [24] to get features combining these modalities. Thus, we first extracted the primary features in each pixel including the absolute differences with the previous frame and the background, the image intensity, and the absolute values $x$- and $y$-derivatives. On the training subset of the smallest 'minimal' split, we then trained a random forest [9] with 5 randomized trees. The training objective was the regression from the appearance of each pixel and its neighborhood to the ground truth density. For each pixel at testtime, the random forest performs a series of simple tests comparing the value of in the particular primary channel at location defined by a particular offset with the particular threshold, while during forest pretraining the number of the channel, the offset and the threshold are randomized. Given the pretrained forest, each pixel $p$ gets assigned a vector $x_p$ of dimension equal to the total number of leaves in all trees, with ones corresponding to the leaves in each of the five trees the pixel falls into and zeros otherwise. Finally, to account for the perspective distortion, we multiplied $x_p$ by the square of the depth of the ground plane at $p$ (provided with the sequence). Within each scenario, we allocated one-fifth of the training frames to pick $\lambda$ and the tree depth through validation via the MESA distance.

The quantitative comparison in Table 2, demonstrates the competitiveness of our method.

**Overall comments.** In both sets of experiments, we tried two strategies for setting $\sigma$ (kernel width in the definition of the ground truth densities): setting $\sigma = 0$ (effectively, the ground truth is then a sum of delta-functions), and setting $\sigma = 4$ (roughly comparable with object half-size in both experiments). In the first case (cells) both strategies gave almost the same results for all $N$, highlighting the insensitivity of our approach to the choice of $\sigma$ (see also Figure 3 on that). The results in Table 1 is for $\sigma = 0$. In the second case (pedestrians), $\sigma = 4$ had an edge over $\sigma = 0$, and the results in Table 2 are for that value.

At train time, we observed that the cutting plane algorithm converged in a few dozen iterations (less than 100 for our choice $\epsilon = 0.01$). The use of a general-purpose quadratic solver [2] meant that the training times were considerable (from several seconds to few hours depending on the value of $\lambda$ and the size of the training set). We anticipate a big reduction in training time for the purpose-built solver. At test time, our approach introduces virtually no time overhead over feature extraction. E.g. in the case of pedestrians, one can store the value $w_t$ computed during learning at each leaf $t$ in each tree, so that counting would require simply "pushing" each pixel down the forest, and summing the resulting $w_t$ from the obtained leaves. This can be done in real-time [30].

## 4 Conclusion

We have presented the general framework for learning to count objects in images. While our ultimate goal is the counting accuracy over the entire image, during the learning our approach is optimizing the loss based on the MESA-distance. This loss involves counting accuracy over multiple subarrays of the entire image (and not only the entire image itself). We demonstrate that given limited amount of training data, such an approach achieves much higher accuracy than optimizing the counting accuracy over the entire image directly (counting-by-regression). At the same time, the fact that we avoid the hard problem of detecting and discerning individual object instances, gives our approach an edge over the counting-by-detection method in our experiments.

**Acknowledgements**. This work is suppoted by EU ERC grant VisRec no. 228180. V. Lempitsky is also supported by Microsoft Research projects in Russia. We thank Prof. Jiri Matas (CTU Prague) for suggesting the *detection+correction* baseline.

# References

[1] http://www.robots.ox.ac.uk/%7Evgg/research/counting/index.html.

[2] The MOSEK optimization software. http://www.mosek.com/.

[3] N. Ahuja and S. Todorovic. Extracting texels in 2.1d natural textures. *ICCV*, pp. 1–8, 2007.

[4] S. An, P. Peursum, W. Liu, and S. Venkatesh. Efficient algorithms for subwindow search in object detection and localization. *CVPR*, pp. 264–271, 2009.

[5] D. Anoraganingrum. Cell segmentation with median filter and mathematical morphology operation. *Image Analysis and Processing, International Conference on*, 0:1043, 1999.

[6] O. Barinova, V. Lempitsky, and P. Kohli. On the detection of multiple object instances using Hough transforms. *CVPR*, 2010.

[7] J. L. Bentley. Programming pearls: Algorithm design techniques. *Comm. ACM*, 27(9):865–871, 1984.

[8] J. L. Bentley. Programming pearls: Perspective on performance. *Comm. ACM*, 27(11):1087–1092, 1984.

[9] L. Breiman. Random forests. *Machine Learning*, 45(1):5–32, 2001.

[10] A. B. Chan, Z.-S. J. Liang, and N. Vasconcelos. Privacy preserving crowd monitoring: Counting people without people models or tracking. *CVPR*, 2008.

[11] S.-Y. Cho, T. W. S. Chow, and C.-T. Leung. A neural-based crowd estimation by hybrid global learning algorithm. *IEEE Transactions on Systems, Man, and Cybernetics, Part B*, 29(4):535–541, 1999.

[12] C. Desai, D. Ramanan, and C. Fowlkes. Discriminative models for multi-class object layout. *ICCV*, 2009.

[13] X. Descombes, R. Minlos, and E. Zhizhina. Object extraction using a stochastic birth-and-death dynamics in continuum. *Journal of Mathematical Imaging and Vision*, 33(3):347–359, 2009.

[14] L. Dong, V. Parameswaran, V. Ramesh, and I. Zoghlami. Fast crowd segmentation using shape indexing. *ICCV*, pp. 1–8, 2007.

[15] M. Everingham, L. Van Gool, C. K. I. Williams, J. Winn, and A. Zisserman. The PASCAL Visual Object Classes Challenge 2009 (VOC2009) Results. http://pascallin.ecs.soton.ac.uk/challenges/VOC/voc2009/workshop/index.html.

[16] T. Joachims, T. Finley, and C.-N. J. Yu. Cutting-plane training of structural svms. *Machine Learning*, 77(1):27–59, 2009.

[17] D. Kong, D. Gray, and H. Tao. A viewpoint invariant approach for crowd counting. *ICPR (3)*, pp. 1187–1190, 2006.

[18] P. D. Kovesi. MATLAB and Octave functions for computer vision and image processing. School of Computer Science & Software Engineering, The University of Western Australia. Available from: http://www.csse.uwa.edu.au/~pk/research/matlabfns/.

[19] A. Lehmussola, P. Ruusuvuori, J. Selinummi, H. Huttunen, and O. Yli-Harja. Computational framework for simulating fluorescence microscope images with cell populations. *IEEE Trans. Med. Imaging*, 26(7):1010–1016, 2007.

[20] B. Leibe, A. Leonardis, and B. Schiele. Robust object detection with interleaved categorization and segmentation. *International Journal of Computer Vision*, 77(1-3):259–289, 2008.

[21] D. G. Lowe. Distinctive image features from scale-invariant keypoints. *International Journal of Computer Vision*, 60(2):91–110, 2004.

[22] A. N. Marana, S. A. Velastin, L. F. Costa, and R. A. Lotufo. Estimation of crowd density using image processing. *Image Processing for Security Applications*, pp. 1–8, 1997.

[23] J. Massey, Frank J. The kolmogorov-smirnov test for goodness of fit. *Journal of the American Statistical Association*, 46(253):68–78, 1951.

[24] F. Moosmann, B. Triggs, and F. Jurie. Fast discriminative visual codebooks using randomized clustering forests. *NIPS*, pp. 985–992, 2006.

[25] S. K. Nath, K. Palaniappan, and F. Bunyak. Cell segmentation using coupled level sets and graph-vertex coloring. *MICCAI (1)*, pp. 101–108, 2006.

[26] T. W. Nattkemper, H. Wersing, W. Schubert, and H. Ritter. A neural network architecture for automatic segmentation of fluorescence micrographs. *Neurocomputing*, 48(1-4):357–367, 2002.

[27] V. Rabaud and S. Belongie. Counting crowded moving objects. *CVPR (1)*, pp. 705–711, 2006.

[28] D. Ryan, S. Denman, C. Fookes, and S. Sridharan. Crowd counting using multiple local features. *DICTA '09: Proceedings of the 2009 Digital Image Computing: Techniques and Applications*, pp. 81–88, 2009.

[29] J. Selinummi, J. Seppala, O. Yli-Harja, and J. A. Puhakka. Software for quantification of labeled bacteria from digital microscope images by automated image analysis. *Biotechniques*, 39(6):859–63, 2005.

[30] T. Sharp. Implementing decision trees and forests on a GPU. *ECCV (4)*, pp. 595–608, 2008.

[31] H. Tamaki and T. Tokuyama. Algorithms for the maxium subarray problem based on matrix multiplication. *SODA*, pp. 446–452, 1998.

[32] A. Vedaldi and B. Fulkerson. VLFeat: An open and portable library of computer vision algorithms. http://www.vlfeat.org/, 2008.

[33] B. Wu, R. Nevatia, and Y. Li. Segmentation of multiple, partially occluded objects by grouping, merging, assigning part detection responses. *CVPR*, 2008.

[34] T. Zhao and R. Nevatia. Bayesian human segmentation in crowded situations. *CVPR (2)*, pp. 459–466, 2003.

